# Continuous Relaxations for Discrete Hamiltonian Monte Carlo

**Yichuan Zhang, Charles Sutton, Amos Storkey**
School of Informatics
University of Edinburgh
United Kingdom
`Y.Zhang-60@sms.ed.ac.uk,`
`csutton@inf.ed.ac.uk,`
`a.storkey@ed.ac.uk`

**Zoubin Ghahramani**
Department of Engineering
University of Cambridge
United Kingdom
`zoubin@eng.cam.ac.uk`

## Abstract

Continuous relaxations play an important role in discrete optimization, but have not seen much use in approximate probabilistic inference. Here we show that a general form of the Gaussian Integral Trick makes it possible to transform a wide class of discrete variable undirected models into fully continuous systems. The continuous representation allows the use of gradient-based Hamiltonian Monte Carlo for inference, results in new ways of estimating normalization constants (partition functions), and in general opens up a number of new avenues for inference in difficult discrete systems. We demonstrate some of these continuous relaxation inference algorithms on a number of illustrative problems.

## 1 Introduction

Discrete undirected graphical models have seen wide use in natural language processing [11, 24] and computer vision [19]. Although sophisticated inference algorithms exist for these models, including both exact algorithms and variational approximations, it has proven more difficult to develop discrete Markov chain Monte Carlo (MCMC) methods. Despite much work and many recent advances [3], the most commonly used MCMC methods in practice for discrete models are based on Metropolis-Hastings, the effectiveness of which is strongly dependent on the choice of proposal distribution.

An appealing idea is to relax the constraint that the random variables of interest take integral values. This is inspired by optimization methods such as linear program relaxation. Continuous problems are appealing because the gradient is on your side: Unlike discrete probability mass functions, in the continuous setting, densities have derivatives, contours, and curvature that can be used to inform sampling algorithms [6, 16, 18, 20, 27]. For this reason, continuous relaxations are widespread in combinatorial optimization, and likewise a major appeal of variational methods is that they convert discrete inference problems into continuous optimization problems. Comparatively speaking, relaxations in an MCMC setting have been generally overlooked.

In this paper we provide a method for relaxing a discrete model into a continuous one, using a technique from statistical physics that Hertz et al. [8] call the "Gaussian integral trick," and that we present in a more general form than is typical. This trick is also known as the Hubbard-Stratonovich transform [10]. Starting with a discrete Markov random field (MRF), the trick introduces an auxiliary Gaussian variable in such a way that the discrete dependencies cancel out. This allows the discrete variables to be summed away, leaving a continuous problem.

The continuous representation allows the use of gradient-based Hamiltonian Monte Carlo for inference, highlights an equivalence between Boltzmann machines and the Gaussian-Bernoulli harmonium model [25], and in general opens up a number of new avenues for inference in difficult discrete

systems. On synthetic problems and a real world problem in text processing, we show that HMC in the continuous relaxation can be much more accurate than standard MCMC methods on the discrete distribution.

The only previous work of which we are aware that uses the Gaussian integral trick for inference in graphical models is Martens and Sutskever [12]. They use the trick to transform an arbitrary MRF into an equivalent restricted Boltzmann machine (RBM), on which they then do block Gibbs sampling. They show that this transformation is useful when each block Gibbs step can be performed in parallel. However, unlike the current work, they do not sum out the discrete variables, so they do not perform a full continuous relaxation.

## 2 Background

Consider an undirected graphical model over random vectors $\mathbf{t} = (t_1, t_2, \ldots t_M)$ where each $t_i \in \{0, 1, 2, \ldots K_i - 1\}$. We will employ a 1 of $K_i$ representation for each non-binary $t_i$ and concatenate the resulting binary variables into the vector $\mathbf{s} = (s_1 \ldots s_N)$. We will also focus on pairwise models over a graph $G = (V, E)$ where $V = \{1, 2, \ldots N\}$. Every discrete undirected model can be converted into a pairwise model at the cost of expanding the state space. The undirected pairwise graphical model can be written in the form

$$p(\mathbf{s}) = \frac{1}{Z} \prod_{(i,j) \in G} \exp(-E_{ij}(s_i, s_j)) \tag{1}$$

where $Z$ is a normalisation term, and is a sum over all valid states of $(s_1, s_2, \ldots, s_N)$ that comply with the 1 of $K_i$ constraints. Equivalently we can set $E_{ij}(s_i, s_j)$ to be very large when $s_i$ and $s_j$ are derived from the same variable $t_k$ (for some $k$ and $i \neq j$, and expanding $G$ to include $(i, j)$), making the resulting product for the terms that break the 1 of $K_i$ constraints to be exponentially small. Henceforth, without loss of generality, we can consider binary pairwise models, and assume $E$ captures any additional constraints that might apply. Then this model takes the general form of a Boltzmann machine or binary MRF, and can be conveniently rewritten as

$$p(\mathbf{s}) = \frac{1}{Z} \exp \left\{ \mathbf{a}^T \mathbf{s} + \frac{1}{2} \mathbf{s}^T W \mathbf{s} \right\} \tag{2}$$

where $\mathbf{a} \in \mathbb{R}^N$, and $W$, a real symmetric matrix, are the model parameters. The normalization function is

$$Z = \sum_{\mathbf{s}} \exp \left\{ \mathbf{a}^T \mathbf{s} + \frac{1}{2} \mathbf{s}^T W \mathbf{s} \right\}. \tag{3}$$

## 3 Gaussian Integral Trick

Inference in Boltzmann machines (which is equivalent to inference in Ising models) has always been a challenging problem. Typically Markov chain Monte Carlo procedures such as Gibbs sampling have been used, but the high levels of connectivity in Boltzmann machines can cause trouble and result in slow mixing in many situations. Furthermore for *frustrated* systems, such models are highly multimodal [1], often with large potential barriers between the different modes.

In many situations, the Hamiltonian Monte Carlo method has provided a more efficient sampling method for highly coupled systems [17], but is only appropriate in real valued problems. For this reason, we choose to work with a real valued augmentation of the Boltzmann machine using the Gaussian integral trick. The main idea is to introduce a real valued auxiliary vector $\mathbf{x} \in \mathbb{R}^N$ in such a way that the $\mathbf{s}^T W \mathbf{s}$ term from (2) cancels out [8]. We generalise the standard form of the Gaussian integral trick by using the following form for the conditional distribution of the auxiliary variable $\mathbf{x}$:

$$p(\mathbf{x}|\mathbf{s}) = \mathcal{N}(\mathbf{x}; A(W + D)\mathbf{s}, A(W + D)A^T) \tag{4}$$

for any choice of invertible matrix $A$ and any diagonal matrix $D$ for which $W + D$ is positive definite. $\mathcal{N}(\mathbf{x}; \mathbf{m}, \Sigma)$ denotes the Gaussian distribution in $\mathbf{x}$ with mean $\mathbf{m}$ and covariance $\Sigma$. The resulting joint distribution over $\mathbf{x}$ and $\mathbf{s}$ is

$$p(\mathbf{x}, \mathbf{s}) \propto \exp(-\frac{1}{2}(\mathbf{x} - A(W + D)\mathbf{s})^T (A^{-1})^T (W + D)^{-1} A^{-1}(\mathbf{x} - A(W + D)\mathbf{s}) + \frac{1}{2}\mathbf{s}^T W \mathbf{s} + \mathbf{a}^T \mathbf{s}). \tag{5}$$

If $\mathbf{d}$ denotes a vector containing the diagonal elements of $D$, this simplifies to

$$p(\mathbf{x}, \mathbf{s}) \propto \exp\left(-\frac{1}{2}\mathbf{x}^T (A^{-1})^T (W+D)^{-1}A^{-1}\mathbf{x} + \mathbf{s}^T A^{-1}\mathbf{x} + (\mathbf{a} - \frac{1}{2}\mathbf{d})^T \mathbf{s}\right). \tag{6}$$

The key point is that the $\mathbf{s}^T W \mathbf{s}$ term has vanished. We can then marginalise out the $\mathbf{s}$ variables, as they are decoupled from one another in the energy function, and can be summed over independently. Define the vector $\alpha_{\mathbf{x}} = A^{-1}\mathbf{x}$. Then the marginal density is

$$p(\mathbf{x}) \propto \exp\left\{-\frac{1}{2}\mathbf{x}^T A^{-1}(W+D)^{-1}(A^{-1})^T\mathbf{x}\right\} \prod_i \left(1 + \exp\left\{\alpha_{\mathbf{x};i} + a_i - \frac{d_i}{2}\right\}\right). \tag{7}$$

The constant of proportionality in the above equation is $Z^{-1}|2\pi A(W+D)A^T|^{-1/2}$. The distribution $p(\mathbf{x})$ is a mixture of $2^N$ Gaussians, i.e., the Gaussians are $p(\mathbf{x}|\mathbf{s})$ with mixing proportion $p(\mathbf{s})$ for each possible assignment $\mathbf{s}$.

We have now converted the discrete distribution $p(\mathbf{s})$ into a corresponding continuous distribution $p(\mathbf{x})$. To understand the sense in which the two distributions "correspond", consider reconstructing $\mathbf{s}$ using the conditional distribution $p(\mathbf{s}|\mathbf{x})$. First, all of the $s_i$ are independent given $\mathbf{x}$, because $\mathbf{s}$ appears only log-linearly in (6). Using the sigmoid $\sigma(z) = (1 + \exp\{-z\})^{-1}$, this is

$$p(s_i|\mathbf{x}) = \sigma\left(-\alpha_{\mathbf{x};i} - a_i + \frac{d_i}{2}\right)^{1-s_i} \sigma\left(\alpha_{\mathbf{x};i} + a_i - \frac{d_i}{2}\right)^{s_i} \tag{8}$$

Two choices for $A$ are of particular interest because they introduce additional independence relationships into the augmented model. First, if $A = \Lambda^{-\frac{1}{2}}V^T$ for the eigendecomposition $W + D = V\Lambda V^T$, then the result is an undirected bipartite graphical model in the joint space of $(\mathbf{x}, \mathbf{s})$:

$$p(\mathbf{x}, \mathbf{s}) \propto \exp\left(-\frac{1}{2}\mathbf{x}^T\mathbf{x} + \mathbf{s}^T V\Lambda^{\frac{1}{2}}\mathbf{x} + (\mathbf{a} - \frac{1}{2}\mathbf{d})^T\mathbf{s}\right). \tag{9}$$

This is a Gaussian-Bernoulli form of exponential family harmonium [25]. Hence we see that the Gaussian-Bernoulli harmonium is equivalent to a general Boltzmann machine over the discrete variables only. Second, if $A = I$ we get

$$p(\mathbf{x}, \mathbf{s}) = Z^{-1}|2\pi(W+D)|^{-1/2} \exp\left\{\left(\mathbf{a} + \mathbf{x} - \frac{1}{2}\mathbf{d}\right)^T \mathbf{s} - \frac{1}{2}\mathbf{x}^T(W+D)^{-1}\mathbf{x}\right\}, \tag{10}$$

which is of particular interest in that the coupling between $\mathbf{s}$ and $\mathbf{x}$ is one-to-one. A given $x_i$ determines the Bernoulli probabilities for the variable $s_i$, independent of the states of any of the other variables. This yields a marginal density

$$p(\mathbf{x}) = Z^{-1}|2\pi(W+D)|^{-1/2} \exp\left\{-\frac{1}{2}\mathbf{x}^T(W+D)^{-1}\mathbf{x}\right\} \prod_i \left(1 + \exp\left\{a_i + x_i - \frac{d_i}{2}\right\}\right) \tag{11}$$

and a particularly nice set of Bernoulli conditional probabilities

$$p(s_i|\mathbf{x}) = \sigma\left(-a_i - x_i + \frac{d_i}{2}\right)^{1-s_i} \sigma\left(a_i + x_i - \frac{d_i}{2}\right)^{s_i} \tag{12}$$

In this model, the marginal of $p(\mathbf{x})$ is a mixture of Gaussian distributions. Then, conditioned on $\mathbf{x}$, the log odds of $s_i = 1$ is a recentered version of $x_i$, in particular, $x_i - a_i - d_i/2$.

The different versions of the Gaussian integral trick can be compactly summarized by the independence relations that they introduce. All versions of Gaussian integral trick give us that all $s_i$ and $s_j$ are independent given $\mathbf{x}$. If we take $A = \Lambda^{-1/2}V^T$, we additionally get that all $x_i$ and $x_j$ are independent given $\mathbf{s}$. Finally if we instead take $A = I$, we get that $s_i$ and $s_j$ are independent given only $x_i$ and $x_j$. These independence relations are presented graphically in Figure 1.

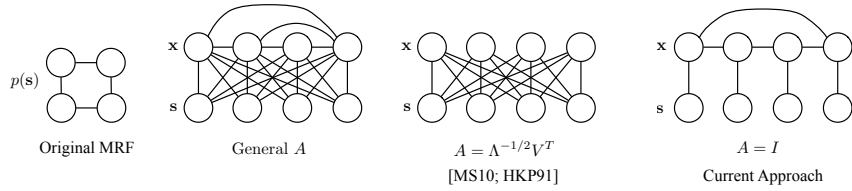

$p(\mathbf{s})$
Original MRF     General $A$     $A = \Lambda^{-1/2}V^T$     $A = I$
                                 [MS10; HKP91]               Current Approach

Figure 1: Graphical depiction of the different versions of the Gaussian integral trick. In all of the models here $s_i \in \{0, 1\}$ while $x_i \in \mathbb{R}$. Notice that when $A = I$ the $\mathbf{x}$ have the same dependence structure as the $\mathbf{s}$ did in the original MRF.

## 3.1 Convexity of Log Density

Because probabilistic inference is NP-hard, it is too much to expect that the continuous transformation will always help. Sometimes difficult discrete distributions will be converted into difficult continuous ones. Experimentally we have noticed that highly frustrated systems typically result in multimodal $p(\mathbf{x})$.

The modes of $p(\mathbf{x})$ are particularly easy to understand if $A = \Lambda^{-1/2}V^T$, because $p(\mathbf{x}|\mathbf{s}) = \mathcal{N}(\mathbf{x}; \Lambda^{1/2}V\mathbf{s}; I)$, that is, the covariance does not depend on $W + D$. Without loss of generality assume that the diagonal of $W$ is 0. Then write $(W + D) = W + cD'$. Interpreting $p(\mathbf{x})$ as a mixture of Gaussians, one for each assignment $\mathbf{s}$, as $c \to \infty$ the Gaussians become farther apart and we get $2^n$ modes, one each at $\Lambda^{1/2}V\mathbf{s}$ for each assignment to binary vector $\mathbf{s}$. If we take a small $c$, however, we can sometimes get fewer modes, and as shown next, we can sometimes even get $\log p(\mathbf{x})$ convex. This is a motivation to make sure that the elements of $D$ are not too large.

In the following proposition we characterize the conditions on $p(\mathbf{s})$ under which the resultant $p(\mathbf{x})$ is log-concave. For any $N \times N$ matrix $M$, let $\lambda_1(M) \geq \ldots \geq \lambda_N(M)$ denote the eigenvalues of $M$. Recall that we have already required that $D$ be chosen so that $W + D$ is positive definite, i.e., $\lambda_N(W + D) > 0$. Then

**Proposition 1.** $p(\mathbf{x})$ *is log-concave if and only if $W + D$ has a narrow spectrum, by which we mean* $\lambda_1(W + D) < 4$.

*Proof.* The Hessian of $\log p(\mathbf{x})$ is easy to compute. It is

$$H_{\mathbf{x}} := \nabla_x^2 \log p(\mathbf{x}) = C_{\mathbf{x}} - (W + D)^{-1} \tag{13}$$

where $C_{\mathbf{x}}$ is a diagonal matrix with elements $c_{ii} = \sigma(-a_i - x_i + \frac{d_i}{2})(1 - \sigma(-a_i - x_i + \frac{d_i}{2}))$. We use the simple eigenvalue inequalities that $\lambda_1(A) + \lambda_N(B) \leq \lambda_1(A + B) \leq \lambda_1(A) + \lambda_1(B)$. If $\lambda_1(W + D) \leq 4$, then

$$\lambda_1(H_{\mathbf{x}}) \leq \lambda_1(C_{\mathbf{x}}) - [\lambda_1(W + D)]^{-1} \leq 0.25 - [\lambda_1(W + D)]^{-1} \leq 0.$$

So $p(\mathbf{x})$ is log-concave. Conversely suppose that $p(\mathbf{x})$ is log-concave. Then

$$0.25 - [\lambda_1(W + D)]^{-1} = \sup_{\mathbf{x}} \lambda_N(C_{\mathbf{x}}) - [\lambda_1(W + D)]^{-1} \leq \sup_{\mathbf{x}} \lambda_1(H_{\mathbf{x}}) \leq 0.$$

So $\lambda_1(W + D) \leq 4$. $\qquad\qquad\square$

## 3.2 MCMC in the Continuous Relaxation

Now we discuss how to perform inference in the augmented distribution resulting from the trick. One simple choice is to focus on the joint density $p(\mathbf{x}, \mathbf{s})$. It is straightforward to generate samples from the conditional distributions $p(\mathbf{x}|\mathbf{s})$ and $p(\mathbf{s}|\mathbf{x})$. Therefore one can sample the joint distribution $p(\mathbf{x}, \mathbf{s})$ in a block Gibbs style that switches sampling between $p(\mathbf{x}|\mathbf{s})$ and $p(\mathbf{s}|\mathbf{x})$. In spite of the simplicity of this method, it has the potential difficulty that it may generate highly correlated samples, due to the coupling between discrete and continuous samples.

To overcome the drawbacks of block Gibbs sampling, we propose running MCMC directly on the marginal $p(\mathbf{x})$. We can efficiently evaluate the unnormalized density of $p(\mathbf{x})$ from (11) up to a

constant, so we can approximately sample from $p(\mathbf{x})$ using MCMC. The derivatives of $\log p(\mathbf{x})$ have a simple form and can be computed at a linear cost of the number of dimension of $\mathbf{x}$.

That suggests the use of Hamiltonian Monte Carlo, an advanced MCMC method that uses gradient information to traverse the continuous space efficiently. We refer to the use of HMC on $p(\mathbf{x})$ as *discrete Hamiltonian Monte Carlo* (DHMC). An important benefit of HMC is that it is more likely than many other Metropolis-Hastings methods to accept a proposed sample that has a large change of log density compared with the current sample.

### 3.3 Estimating Marginal Probabilities

Given a set of samples that are approximately distributed from $p(\mathbf{x})$ we can estimate the marginal distribution over any subset $S_q \subseteq S$ of the discrete variables. This is possible because all of the variables $S_q$ decompose given $\mathbf{x}$. There is no need to generate samples of $\mathbf{s}$ because $p(\mathbf{s}|\mathbf{x})$ is easy to compute. The marginal probability $p(\mathbf{s}_q)$ can be estimated as

$$p(\mathbf{s}_q) \approx \frac{1}{M}\sum_{m=1}^{M} p(S_q|\mathbf{x}^{(m)}) = \frac{1}{M}\sum_{m=1}^{M} \prod_{s_i \in S_q} p(\mathbf{s}_i|\mathbf{x}^{(m)}) \qquad \mathbf{x}^{(m)} \sim p(\mathbf{x})$$

This gives us a Rao-Blackwellized estimate of $p(\mathbf{s}_q)$ without needing to sample $\mathbf{s}$ directly. We will not typically be able to obtain exact samples from p(x), as in general it may be multimodal, but we can obtain approximate samples from an MCMC method.

### 3.4 Normalizing Constants

Because the normalizing factor $Z^{-1}$ of the Boltzmann machine is equal to the probability $p(\mathbf{s} = 0)$, we can estimate the normalizing factor using the technique from the previous section

$$Z^{-1} = p(\mathbf{s} = 0) \approx \frac{1}{M}\sum_{m=1}^{M} p(\mathbf{s} = 0|\mathbf{x}^{(m)}) \quad \mathbf{x}^{(m)} \sim p(\mathbf{x}). \tag{14}$$

Although if we can sample exactly from $p(\mathbf{x})$, this estimator is unbiased, it suffers from two problems. First, because it is an estimator of $Z^{-1}$, as [15] explains, such an estimator may underestimate $Z$ and $\log Z$ in practice. Second, it can suffer a similar problem as the harmonic mean estimator. The non-Gaussian term in $p(\mathbf{x})$ corresponds to $p(\mathbf{s} = 0|\mathbf{x})^{-1}$. In general it is difficult to approximate the expectation of a function $f(x)$ with respect to a distribution $q(x)$ if $f$ is large precisely where $q$ is not. This is potentially the situation with this estimator of $Z$.

We use an alternative estimator using a "mirrored variant" of the importance trick. First, we introduce a distribution $q(\mathbf{x})$. Define $p^*(\mathbf{x}) = Zp(\mathbf{x})$ to be an unnormalized version of $p$. Using the identity $Z^{-1} = Z^{-1}\int d\mathbf{x}\, q(\mathbf{x})$ we have

$$Z^{-1} = Z^{-1}\int d\mathbf{x}\, q(\mathbf{x})\frac{p^*(\mathbf{x})}{p^*(\mathbf{x})} = \int d\mathbf{x}\, \frac{q(\mathbf{x})}{p^*(\mathbf{x})}p(\mathbf{x}),$$

A Monte Carlo estimate of this integral is $\hat{Z}^{-1} = \frac{1}{M}\sum_m \frac{q(\mathbf{x}^{(m)})}{p^*(\mathbf{x}^{(m)})}$ for $\mathbf{x}^{(m)} \sim p(\mathbf{x})$. This estimator reduces to (14) if we take $q(\mathbf{x})$ to be the Gaussian portion of $p(\mathbf{x})$ i.e., $q(\mathbf{x}) = \mathcal{N}(\mathbf{x}; 0, A(W + D)A^T)$. However in practice we have found other choices of $q$ to be much better. Intuitively, the variance of $\hat{Z}$ depends on the ratio of $q(\mathbf{x})/p^*(\mathbf{x})$. If $p(\mathbf{x}) = q(\mathbf{x})$, the variance of this estimator is asymptotically zero. This importance trick is well known in the statistics literature, e.g., [14].

## 4 Related Work

The use of Hamiltonian mechanics in Boltzmann machines and related models (e.g., Ising Models, stochastic Hopfield models) has an interesting history. The Ising model was studied as a model of physical spin systems, and so the dynamics used were typically representative of the physics, with Glauber dynamics [7] being a common model, e.g., [2]. In the context of stochastic neural models, though, there was the potential to examine other dynamics that did not match the standard

physics of spin systems. Hamiltonian dynamics were considered as a neural interaction model [13, 21, 26], but were not applied directly to the Ising model itself, or used for inference. Hamiltonian dynamics were also considered for networks combining excitatory and inhibitory neurons [22]. All these approaches involved developing Hamiltonian neural models, rather than Hamiltonian auxiliary methods for existing models.

The Gaussian integral trick is also known as the Hubbard-Stratonovich transformation in physics[10]. In the area of neural modelling, the Gaussian integral trick was also common for theoretical reasons rather than as a practical augmentation strategy [8]. The Gaussian integral trick formed a critical part of replica-theoretical analysis [8] for phase analysis of spin systems, as it enabled ensemble averaging of the spin components, leading to saddle-point equations in the continuous domain. These theoretical analysis relied on the ensemble randomness of the interaction matrix of the stochastic Hopfield model, and so were not directly relevant in the context of a learnt Boltzmann machine, where the weight matrix has a specific structure.

The Gaussian integral trick relates the general Boltzmann machines to exponential family harmoniums [25], which generalise the restricted Boltzmann machines. The specific Gaussian-Bernoulli harmonium is in common use, but where the real valued variables are visible units and the binary variables are hidden variables [9]. This is quite distinct from the use here where the visible and hidden units are all binary and the Gaussian variables are auxiliary variables.

The only work of which we are aware that uses the Gaussian integral trick for probabilistic inference is that of Martens and Sutskever [12]. This work also considers inference in MRFs, using the special case of the Gaussian integral trick in which $A = \Lambda^{-\frac{1}{2}} V^T$. However, they do not use the full version of the trick, as they do not integrate $\mathbf{s}$ out, so they never obtain a fully continuous problem. Instead they perform inference directly in the resulting harmonium, using block Gibbs sampling alternating between $\mathbf{s}$ and $\mathbf{x}$. On serial computers, they do not find that this expanded representation offers much benefit over performing single-site Gibbs in the discrete space. Indeed they find that the sampler in the augmented model is actually slightly slower than the one in the original discrete space. This is in sharp contrast to our work, because we use a Rao-Blackwellized sampler on the $\mathbf{x}$.

# 5    Results

In this section we evaluate the accuracy of the relaxed sampling algorithms on both synthetic grids and a real-world task. We evaluate both the estimation of node marginal and of the normalisation factor estimation on two synthetic models.

We compare the accuracy of the discrete HMC sampler to Gibbs sampling in the original discrete model $p(\mathbf{s})$ and to block Gibbs sampling the augmented model $p(\mathbf{x}, \mathbf{s})$. We choose the number of MCMC samples so that the total computational time for each method is roughly the same. The Gibbs sampler resamples one node at a time from $p(s_i | \mathbf{s}_{-i})$. The node marginal probability $p(s_i)$ is estimated by the empirical probability of the samples. The normalizing constant is estimated in Chib-style using the Gibbs transition kernel, for more details see [15].

The block Gibbs sampler over $p(\mathbf{x}, \mathbf{s})$ we use is based on [12]. This comparison is designed to evaluate the benefits of summing away the discrete variables. To estimate the node marginals, we use the block Gibbs sampler to generate samples of $\mathbf{x}$ and then apply the Rao-Blackwellized estimators from Sections 3.3 and 3.4. We empirically choose the mirror distribution $q$ as a Gaussian distribution, with mean and variance given by the empirical mean and covariance of the $\mathbf{x}$ samples from MCMC. The samples of $\mathbf{s}$ are simply discarded at the estimation stage.

HMC can generate better samples while a large number of leapfrog steps is used, but this requires much more computation time. For a fixed computational budget, using more leapfrog steps causes fewer samples to be generated, which can also undermine the accuracy of the estimator. So, we empirically pick 5 leapfrog steps and tuning the leapfrog step size so that acceptance rate is around 90%. We use $A = I$ for DHMC. To estimate the marginals $p(s_i)$ and the partition function, we apply the Rao-Blackwellized estimators from Sections 3.3 and 3.4 in the same way as for block Gibbs.

**Synthetic Boltzmann Machines.** We evaluate the performance of the samplers across different types of weight matrices by using synthetically generated models. The idea is to characterize what types of distributions are difficult for each sampler.

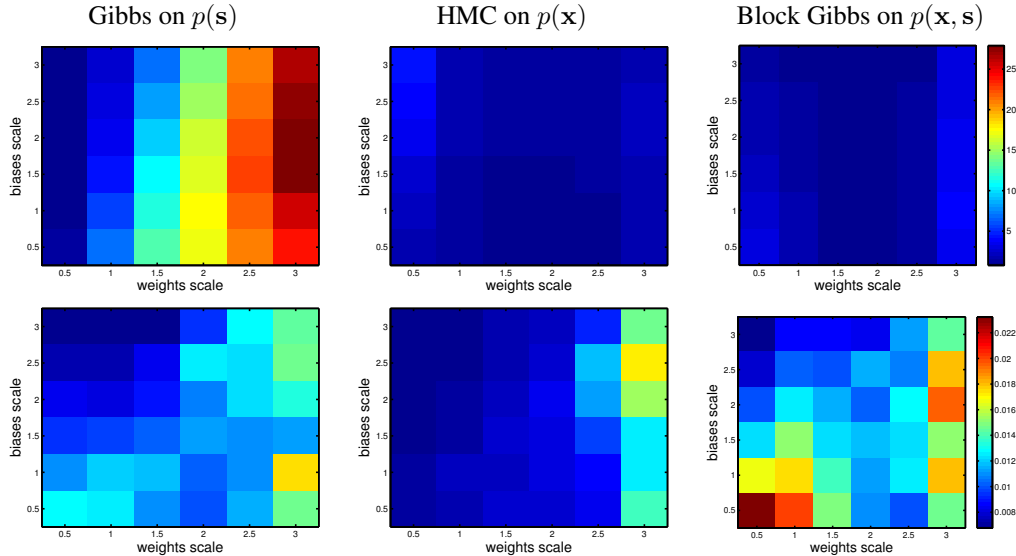

Figure 2: Performance of samplers on synthetic grid-structured Boltzmann machines. The axes show the standard deviations of the distributions used to select the synthetic models. The top row shows error in the normalization constant, while the bottom row shows average error in the single-mode marginal distributions.

We randomly generate $10 \times 10$ grid models over binary variables. We use two different generating processes, a "standard" one and a "frustrated" one. In the standard case, for each node $s_i$, the biases are generated as $a_i \sim c_1 \mathcal{N}(0, 4)$. The weights are generated as $w_{ij} \sim c_2 \mathcal{N}(0, 4)$. The parameters $c_1$ and $c_2$ define the scales of the biases and weights and determine how hard the problem is. In the frustrated case, we shift the weights to make the problem more difficult. We still generate the weights as $w_{ij} \sim c_2 \mathcal{N}(0, 4)$ but now we generate the biases as $a_i \sim c_1 \mathcal{N}(\sum_i w_{ij}, 4)$. This shift in the Gaussian distribution tends to encourage the multimodality of $p(\mathbf{x})$.

We test all three samplers on 36 random graphs from each of the two generating processes, using different values of $c_1$ and $c_2$ for each random graph. Each MCMC method is given 10 runs of 10000 samples with 2000 burn-in samples. We report the MSE of the node marginal estimate and the log normalising constant estimate averaged over 10 runs.

The results are shown in Figure 2 and 3. The axes show $c_1$ and $c_2$, which determine the difficulty of the problem. The higher value in the heat maps means a larger error. On the standard graphs (Figure 2), the DHMC method significantly outperforms both competitors. DHMC beats Gibbs on $p(\mathbf{s})$ at the normalization constant and beats block Gibbs on $p(\mathbf{x}, \mathbf{s})$ at marginal estimation.

The frustrated graphs (Figure 3) are significantly more difficult for DHMC, as expected. All three samplers seem to have trouble in the same area of model space, although DHMC suffers somewhat worse than the other methods in marginal error, while still beating Chib's method for normalization constants. It is noted that in worst scenarios, the weights of the model are very extreme. Examining a few representative graphs seems to indicate that in the regime with large weights, the HMC sampler becomes stuck in a bad mode. We observe that in both cases block Gibbs of $p(\mathbf{x}, \mathbf{s})$ performs roughly the same at marginal estimation as Gibbs on $p(\mathbf{s})$. This is consistent with the results in [12].

**Text data.** We also evaluate the relaxed inference methods on a small text labelling data set. The data are a series of email messages that announce academic seminars [5]. We consider the binary problem of determining whether or not each word in the message is part of the name of the seminar's speaker, so we have one random variable for each token in the message. We use a "skip-chain CRF" [4, 23] model which contains edges between adjacent tokens and also additional edges between any pair of identical capitalized words.

We trained the CRF on a set of 485 messages using belief propagation. We evaluate the performance of different inference methods on inferring the probability distribution over labels on a held out set

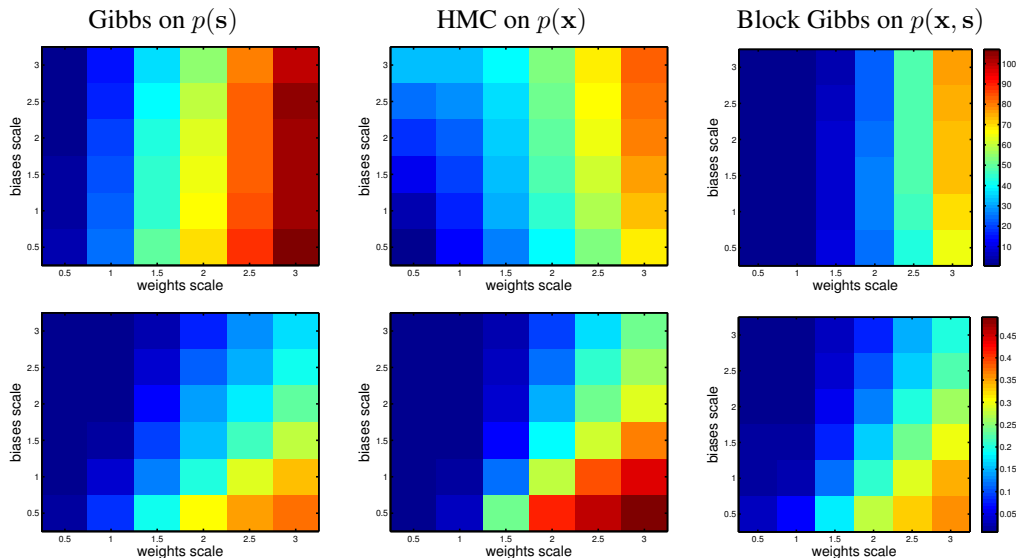

Figure 3: Performance of samplers on a set of highly frustrated grid-structured Boltzmann machines. The axes show the standard deviations of the distributions used to select the synthetic models. The top row shows error in the normalization constant, while the bottom row shows average error in the single-mode marginal distributions.

of messages. Our test set uses a subset of the messages which are small enough that we can run exact inference. The test set contained 75 messages whose length ranged from 50 words to 628 words. We evaluate whether the approximate inference methods match the solution from exact inference.

The accuracy of the three approximate inference methods are shown in Table 1. We see that the HMC sampler is much more accurate than either of the other samplers at estimating single-node marginals. Chib's method and DHMC have roughly the same accuracy on the normalization constant. The block Gibbs sampler yields both worse estimates of the marginals and a significantly worse estimate of the normalization constant.

| | Gibbs $p(\mathbf{s})$ | DHMC $p(\mathbf{x})$ | Block Gibbs $p(\mathbf{s}, \mathbf{x})$ |
|---|---|---|---|
| RMSE (node marginal) | 0.2346 | **0.1619** | 0.2251 |
| RSME (log normalizing constant) | **3.3041** | 3.3171 | 12.9685 |

Table 1: Root mean squared error of single site marginal and normalising constant against the ground truth computed by the variable elimination algorithm

## 6  Conclusion

We have provided a general strategy for approximate inference based on relaxing discrete distributions into continuous ones using the classical Gaussian integral trick. We described a continuum of different versions of the trick that have different properties. Although we illustrated the benefits of the continuous setting by using Hamiltonian Monte Carlo, in future work other inference methods such as elliptical slice sampling or more advanced HMC methods may prove superior. We hope that this work might open the door to a larger space of interesting relaxations for approximate inference.

**Acknowledgments**

We thank Iain Murray for helpful comments, Max Welling for introducing ZG to the GIT and Peter Sollich for bringing the paper by Hubbard to our attention. This work was supported by the Engineering and Physical Sciences Research Council [grant numbers EP/I036575/1 and EP/J00104X/1].

# References

[1] D. J. Amit. *Modeling Brain Function*. Cambridge University Press, 1989.

[2] A. Coolen, S. Laughton, and D. Sherrington. Modern analytic techniques to solve the dynamics of recurrent neural networks. In *Advances in Neural Information Processing Systems 8 (NIPS95)*, 1996.

[3] S. Ermon, C. P. Gomes, A. Sabharwal, and B. Selman. Accelerated adaptive Markov chain for partition function computation. In J. Shawe-Taylor, R. Zemel, P. Bartlett, F. Pereira, and K. Weinberger, editors, *Advances in Neural Information Processing Systems 24*, pages 2744–2752. 2011.

[4] J. Finkel, T. Grenager, and C. D. Manning. Incorporating non-local information into information extraction systems by Gibbs sampling. In *Annual Meeting of the Association for Computational Linguistics (ACL)*, 2005.

[5] D. Frietag and A. McCallum. Information extraction with HMMs and shrinkage. In *AAAI Workshop on Machine Learning for Information Extraction*, 1999.

[6] M. Girolami, B. Calderhead, and S. A. Chin. Riemannian manifold Hamiltonian Monte Carlo. *Journal of the Royal Statistical Society, B*, 73(2):1–37, 2011.

[7] R. J. Glauber. Time-dependent statistics of the Ising model. *J. Math. Phys.*, 4:294–307, 1963.

[8] J. Hertz, A. Krogh, and R. G. Palmer. *Introduction to the Theory of Neural Computation*. Perseus Books, 1991.

[9] G. E. Hinton and R. R. Salakhutdinov. Reducing the dimensionality of data with neural networks. *Science*, 313:504–507, 2006.

[10] J. Hubbard. Calculation of partition functions. *Phys. Rev. Lett.*, 3:77–78, Jul 1959. doi: 10.1103/PhysRevLett.3.77. URL http://link.aps.org/doi/10.1103/PhysRevLett.3.77.

[11] M. Johnson, T. Griffiths, and S. Goldwater. Bayesian inference for PCFGs via Markov chain Monte Carlo. In *HLT/NAACL*, 2007.

[12] J. Martens and I. Sutskever. Parallelizable sampling of Markov random fields. In *Conference on Artificial Intelligence and Statistics (AISTATS)*, 2010.

[13] R. V. Mendes and J. T. Duarte. Vector fields and neural networks. *Complex Systems*, 6:21–30, 1992.

[14] J. Møller, A. N. Pettitt, R. Reeves, and K. K. Berthelsen. An efficient Markov chain Monte Carlo method for distributions with intractable normalising constants. *Biometrika*, 93(2):pp. 451–458, 2006.

[15] I. Murray and R. Salakhutdinov. Evaluating probabilities under high-dimensional latent variable models. In D. Koller, D. Schuurmans, Y. Bengio, and L. Bottou, editors, *Advances in Neural Information Processing Systems 21*, pages 1137–1144. 2009.

[16] I. Murray, R. P. Adams, and D. J. MacKay. Elliptical slice sampling. *JMLR: W&CP*, 9:541–548, 2010.

[17] R. Neal. *Bayesian Learning for Neural Networks*. PhD thesis, Computer Science, University of Toronto, 1995.

[18] R. M. Neal. MCMC using Hamiltonian dynamics. In S. Brooks, A. Gelman, G. Jones, and X.-L. Meng, editors, *Handbook of Markov Chain Monte Carlo*. Chapman & Hall / CRC Press, 2010.

[19] S. Nowozin and C. H. Lampert. Structured prediction and learning in computer vision. *Foundations and Trends in Computer Graphics and Vision*, 6(3-4), 2011.

[20] Y. Qi and T. P. Minka. Hessian-based Markov chain Monte-Carlo algorithms. In *First Cape Cod Workshop on Monte Carlo Methods*, September 2002.

[21] U. Ramacher. The Hamiltonian approach to neural networks dynamics. In *Proc. IEEE JCNN*, volume 3, 1991.

[22] H. S. Seung, T. J. Richardson, J. C. Lagarias, and J. J. Hopfield. Minimax and Hamiltonian dynamics of excitatory-inhibitory networks. In *Advances in Neural Information Processing Systems 10 (NIPS97)*, 1998.

[23] C. Sutton and A. McCallum. Collective segmentation and labeling of distant entities in information extraction. In *ICML Workshop on Statistical Relational Learning and Its Connections to Other Fields*, 2004.

[24] C. Sutton and A. McCallum. An Introduction to Conditional Random Fields. *Foundations and Trends in Machine Learning*, 4(4), 2012.

[25] M. Welling, M. Rosen-Zvi, and G. Hinton. Exponential family harmoniums with an application to information retrieval. In *Advances in Neural Information Processing 17*, 2004.

[26] P. D. Wilde. Class of Hamiltonian neural networks. *Physical Review E*, 2:1392–1396, 47.

[27] Y. Zhang and C. Sutton. Quasi-Newton Markov chain Monte Carlo. In *Advances in Neural Information Processing Systems (NIPS)*, 2011.

